# Probabilistic Modeling of Dependencies Among Visual Short-Term Memory Representations

**A. Emin Orhan**     **Robert A. Jacobs**
Department of Brain & Cognitive Sciences
University of Rochester
Rochester, NY 14627
{eorhan,robbie}@bcs.rochester.edu

## Abstract

Extensive evidence suggests that items are not encoded independently in visual short-term memory (VSTM). However, previous research has not quantitatively considered how the encoding of an item influences the encoding of other items. Here, we model the dependencies among VSTM representations using a multivariate Gaussian distribution with a stimulus-dependent mean and covariance matrix. We report the results of an experiment designed to determine the specific form of the stimulus-dependence of the mean and the covariance matrix. We find that the magnitude of the covariance between the representations of two items is a monotonically decreasing function of the difference between the items' feature values, similar to a Gaussian process with a distance-dependent, stationary kernel function. We further show that this type of covariance function can be explained as a natural consequence of encoding multiple stimuli in a population of neurons with correlated responses.

## 1   Introduction

In each trial of a standard visual short-term memory (VSTM) experiment (e.g. [1,2]), subjects are first presented with a display containing multiple items with simple features (e.g. colored squares) for a brief duration and then, after a delay interval, their memory for the feature value of one of the items is probed using either a recognition or a recall task. Let $\mathbf{s} = [s_1, s_2, \ldots, s_N]^T$ denote the feature values of the $N$ items in the display on a given trial. In this paper, our goal is to provide a quantitative description of the content of a subject's visual memory for the display after the delay interval. That is, we want to characterize a subject's belief state about $\mathbf{s}$.

We suggest that a subject's belief state can be expressed as a random variable $\hat{\mathbf{s}} = [\hat{s}_1, \hat{s}_2, \ldots, \hat{s}_N]^T$ that depends on the actual stimuli $\mathbf{s}$: $\hat{\mathbf{s}} = \hat{\mathbf{s}}(\mathbf{s})$. Consequently, we seek a suitable joint probability model $p(\hat{\mathbf{s}})$ that can adequately capture the content of a subject's memory of the display. We note that most research on VSTM is concerned with characterizing how subjects encode a single item in VSTM (for instance, the precision with which a single item can be encoded [1,2]) and, thus, does not consider the joint encoding of multiple items. In particular, we are not aware of any previous work attempting to experimentally probe and characterize exactly how the encoding of an item influences the encoding of other items, i.e. the joint probability distribution $p(\hat{s}_1, \hat{s}_2, \ldots, \hat{s}_N)$.

A simple (perhaps simplistic) suggestion is to assume that the encoding of an item does not influence the encoding of other items, i.e. the feature values of different items are represented independently in VSTM. If so, the joint probability distribution factorizes as $p(\hat{s}_1, \hat{s}_2, \ldots, \hat{s}_N) = p(\hat{s}_1)p(\hat{s}_2)\ldots p(\hat{s}_N)$. However, there is now extensive evidence against this simple model [3,4,5,6].

## 2 A Gaussian process model

We consider an alternative model for $p(\hat{s}_1, \hat{s}_2, \ldots, \hat{s}_N)$ that allows for dependencies among representations of different items in VSTM. We model $p(\hat{s}_1, \hat{s}_2, \ldots, \hat{s}_N)$ as an $N$-dimensional multivariate Gaussian distribution with mean $\mathbf{m}(\mathbf{s})$ and full covariance matrix $\Sigma(\mathbf{s})$, both of which depend on the actual stimuli $\mathbf{s}$ appearing in a display. This model assumes that only pairwise (or second-order) correlations exist between the representations of different items. Although more complex models incorporating higher-order dependencies between the representations of items in VSTM can be considered, it would be difficult to experimentally determine the parameters of these models. Below we show how the parameters of the multivariate Gaussian model, $\mathbf{m}(\mathbf{s})$ and $\Sigma(\mathbf{s})$, can be experimentally determined from standard VSTM tasks with minor modifications.

Importantly, we emphasize the dependence of $\mathbf{m}(\mathbf{s})$ and $\Sigma(\mathbf{s})$ on the actual stimuli $\mathbf{s}$. This is to allow for the possibility that subjects might encode stimuli with different similarity relations differently. For instance (and to foreshadow our experimental results), if the items in a display have similar feature values, one might reasonably expect there to be large dependencies among the representations of these items. Conversely, the correlations among the representations of items might be smaller if the items in a display are dissimilar. These two cases would imply different covariance matrices $\Sigma$, hence the dependence of $\Sigma$ (and $\mathbf{m}$) on $\mathbf{s}$.

Determining the properties of the covariance matrix $\Sigma(\mathbf{s})$ is, in a sense, similar to finding an appropriate kernel for a given dataset in the Gaussian process framework [7]. In Gaussian processes, one expresses the covariance matrix in the form $\Sigma_{ij} = k(s_i, s_j)$ using a parametrized kernel function $k$. Then one can ask various questions about the kernel function: What kind of kernel function explains the given dataset best, a stationary kernel function that only depends on $|s_i - s_j|$ or a more general, non-stationary kernel? What parameter values of the chosen kernel (e.g. the scale length parameter for a squared exponential type kernel) explain the dataset best? We ask similar questions about our stimulus-dependent covariance matrix $\Sigma(\mathbf{s})$: Does the covariance between VSTM representations of two stimuli depend only on the absolute difference between their feature values, $|s_i - s_j|$, or is the relationship non-stationary and more complex? If the covariance function is stationary, what is its scale length (how quickly does the covariance dissipate with distance)? In Section 3, we address these questions experimentally.

**Why does providing an appropriate context improve memory?**

Modeling subjects' VSTM representations of multiple items as a joint probability distribution allows us to explain an intriguing finding by Jiang, Olson and Chun [3] in an elegant way. We first describe the finding, and then show how to explain this result within our framework.

Jiang et al. [3] showed that relations between items in a display, as well as items' individual characteristics, are encoded in VSTM. In their Experiment 1, they briefly presented displays consisting of colored squares to subjects. There were two test or probe conditions. In the single probe condition, only one of the squares (called the target probe) reappeared, either with the same color as in the original display, or with a different color. In the minimal color change condition, the target probe (again with the same color or with a different color) reappeared together with distracter probes which always had the same colors as in the original display. In both conditions, subjects decided whether a color change occurred in the target probe. Jiang et al. [3] found that subjects' performances were significantly better in the minimal color change condition than in the single probe condition. This result suggests that the color for the target square was not encoded independently of the colors of the distracter squares because if the target color was encoded independently then subjects would have shown identical performances regardless of whether distractor squares were present (minimal color change condition) or absent (single probe condition). In Experiment 2 of [3], a similar result was obtained for location memory: location memory for a target was better in the minimal change condition than in the single probe condition or in a maximal change condition where all distracters were presented but at different locations than their original locations.

These results are easy to understand in terms of our joint probability model for item memories, $p(\hat{\mathbf{s}})$. Intuitively, the single probe condition taps the marginal probability of the memory for the target item, $p(\hat{s}_t)$, where $t$ represents the index of the target item. In contrast, the minimal color change condition taps the conditional probability of the memory for the target given the memories for the distracters, $p(\hat{s}_t | \hat{\mathbf{s}}_{-t} = \mathbf{s}_{-t})$ where $-t$ represents the indices of the distracter items, because the actual dis-

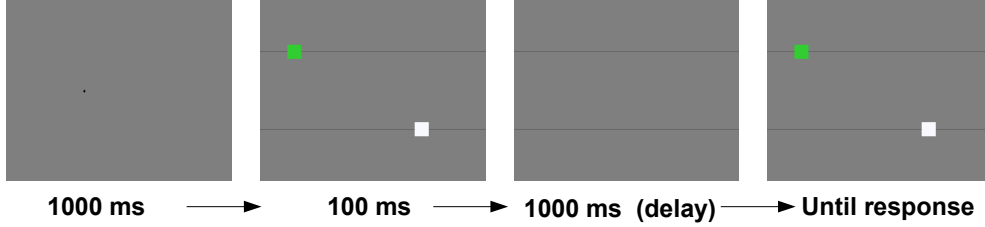

**1000 ms** ⟶ **100 ms** ⟶ **1000 ms (delay)** ⟶ **Until response**

Figure 1: The sequence of events on a single trial of the experiment with $N = 2$.

tracters $\mathbf{s}_{-t}$ are shown during test. If the target probe has high probability under these distributions, then the subject will be more likely to respond 'no-change', whereas if it has low probability, then the subject will be more likely to respond 'change'. If the items are represented independently in VSTM, the marginal and conditional distributions are the same; i.e. $p(\hat{s}_t) = p(\hat{s}_t|\hat{\mathbf{s}}_{-t})$. Hence, the independent-representation assumption predicts that there should be no difference in subjects' performances in the single probe and minimal color change conditions. The significant differences in subjects' performances between these conditions observed in [3] provides evidence against the independence assumption.

It is also easy to understand why subjects performed better in the minimal color change condition than in the single probe condition. The conditional distribution $p(\hat{s}_t|\hat{\mathbf{s}}_{-t})$ is, in general, a lower-variance distribution than the marginal distribution $p(\hat{s}_t)$. Although this is not exclusively true for the Gaussian distribution, it can analytically be proven in the Gaussian case. If $p(\hat{\mathbf{s}})$ is modeled as an $N$-dimensional multivariate Gaussian distribution:

$$\hat{\mathbf{s}} = [\hat{s}_t, \hat{\mathbf{s}}_{-t}]^T \sim \mathcal{N}([a, b]^T, [A, \ C; \ C^T, \ B]) \tag{1}$$

(where the covariance matrix is written using Matlab notation), then the conditional distribution $p(\hat{s}_t|\hat{\mathbf{s}}_{-t})$ has mean $a + CB^{-1}(\hat{\mathbf{s}}_{-t} - b)$ and variance $A - CB^{-1}C^T$, whereas the marginal distribution $p(\hat{s}_t)$ has mean $a$ and variance $A$ which is always greater than $A - CB^{-1}C^T$. [As an aside, note that when the distracter probes are different from the mean of the memories for distracters, i.e. $\hat{\mathbf{s}}_{-t} \neq b$, the conditional distribution $p(\hat{s}_t|\hat{\mathbf{s}}_{-t})$ is biased away from $a$, explaining the poorer performance in the maximal change condition than in the single probe condition.]

## 3 Experiments

We conducted two VSTM recall experiments to determine the properties of $\mathbf{m}(\mathbf{s})$ and $\Sigma(\mathbf{s})$. The experiments used position along a horizontal line as the relevant feature to be remembered.

**Procedure:** Each trial began with the display of a fixation cross at a random location within an approximately $12° \times 16°$ region of the screen for 1 second. Subjects were then presented with a number of colored squares ($N = 2$ or $N = 3$ squares in separate experiments) on linearly spaced dark and thin horizontal lines for 100 ms. After a delay interval of 1 second, a probe screen was presented. Initially, the probe screen contained only the horizontal lines. Subjects were asked to use the computer mouse to indicate their estimate of the horizontal location of each of the colored squares presented on that trial. We note that this is a novelty of our experimental task, since in most other VSTM tasks, only one of the items is probed and the subject is asked to report the content of their memory associated with the probed item. Requiring subjects to indicate the feature values of all presented items allows us to study the dependencies between the memories for different items. Subjects were allowed to adjust their estimates as many times as they wished. When they were satisfied with their estimates, they proceeded to the next trial by pressing the space bar. Figure 1 shows the sequence of events on a single trial of the experiment with $N = 2$.

To study the dependence of $\mathbf{m}(\mathbf{s})$ and $\Sigma(\mathbf{s})$ on the horizontal locations of the squares $\mathbf{s} = [s_1, s_2, \ldots, s_N]^T$, we used different values of $\mathbf{s}$ on different trials. We call each different $\mathbf{s}$ a particular 'display configuration'. To cover a range of possible display configurations, we selected uniformly-spaced points along the horizontal dimension, considered all possible combinations of these points (e.g. item 1 is at horizontal location $s_1$ and item 2 is at location $s_2$), and then added a small amount of jitter to each combination. In the experiment with two items, 6 points were selected along the horizontal dimension, and thus there were 36 ($6 \times 6$) different display configurations. In the experiment with three items, 3 points were selected along the horizontal dimension, meaning that 27 ($3 \times 3 \times 3$) configurations were used.

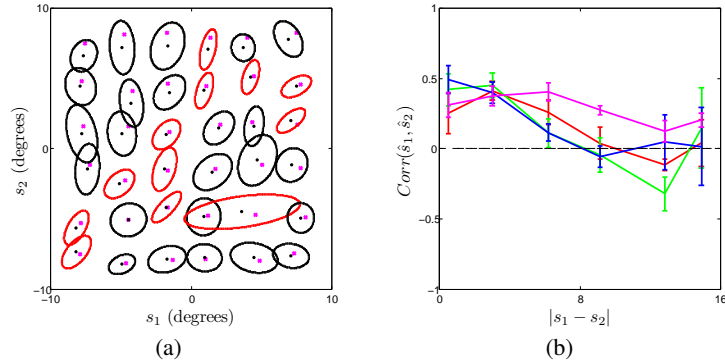

Figure 2: (a) Results for subject RD. The actual display configurations $\mathbf{s}$ are represented by magenta dots, the estimated means based on the subject's responses are represented by black dots and the estimated covariances are represented by contours (with red contours representing $\Sigma(\mathbf{s})$ for which the two dimensions were significantly correlated at the $p < 0.05$ level). (b) Results for all 4 subjects. The graph plots the mean correlation coefficients (and standard errors of the means) as a function of $|s_1 - s_2|$. Each color corresponds to a different subject.

Furthermore, since $\mathbf{m}(\mathbf{s})$ and $\Sigma(\mathbf{s})$ cannot be reliably estimated from a single trial, we presented the same configuration $\mathbf{s}$ a number of times and collected the subject's response each time. We then estimated $\mathbf{m}(\mathbf{s})$ and $\Sigma(\mathbf{s})$ for a particular configuration $\mathbf{s}$ by fitting an $N$-dimensional Gaussian distribution to the subject's responses for the corresponding $\mathbf{s}$. We thus assume that when a particular configuration $\mathbf{s}$ is presented in different trials, the subject forms and makes use of (i.e. samples from) roughly the same VSTM representation $p(\hat{\mathbf{s}}) = \mathcal{N}(\mathbf{m}(\mathbf{s}), \Sigma(\mathbf{s}))$ in reporting the contents of their memory. In the experiment with $N = 2$, each of the 36 configurations was presented 24 times (yielding a total of 864 trials) and in the experiment with $N = 3$, each of the 27 configurations was presented 26 times (yielding a total of 702 trials), randomly interleaved. Subjects participating in the same experiment (either two or three items) saw the same set of display configurations.

**Participants:** 8 naive subjects participated in the experiments (4 in each experiment). All subjects had normal or corrected-to-normal vision, and they were compensated at a rate of \$10 per hour. For both set sizes, subjects completed the experiment in two sessions.

**Results:** We first present the results for the experiment with $N = 2$. Figure 2a shows the results for a representative subject (subject RD). In this graph, the actual display configurations $\mathbf{s}$ are represented by magenta dots, the estimated means $\mathbf{m}(\mathbf{s})$ based on the subject's responses are represented by black dots and the estimated covariances $\Sigma(\mathbf{s})$ are represented by contours (red contours represent $\Sigma(\mathbf{s})$ for which the two dimensions were significantly ($p < 0.05$) correlated). For this particular subject, $p(\hat{s}_1, \hat{s}_2)$ exhibited a significant correlation for 12 of 36 configurations. In all these cases, correlations were positive, meaning that when the subject made an error in a given direction for one of the items, s/he was likely to make an error in the same direction for the other item. This tendency was strongest when items were at similar horizontal positions [e.g. distributions are more likely to exhibit significant correlations for display configurations close to the main diagonal ($s_1 = s_2$)].

Figure 2b shows results for all 4 subjects. This graph plots the correlation coefficients for subjects' position estimates as a function of the absolute differences in items' positions ($|s_1 - s_2|$). In this graph, configurations were divided into 6 equal-length bins according to their $|s_1 - s_2|$ values, and the correlations shown are the mean correlation coefficients (and standard errors of the means) for each bin. Clearly, the correlations decrease with increasing $|s_1 - s_2|$. Correlations differed significantly across different bins (one-way ANOVA: $p < .05$ for all but one subject, as well as for combined data from all subjects). One might consider this graph as representing a stationary kernel function that specifies how the covariance between the memory representations of two items changes as a function of the distance $|s_1 - s_2|$ between their feature values. However, as can be observed from Figure 2a, the experimental kernel function that characterizes the dependencies between the VSTM representations of different items is not perfectly stationary. Additional analyses (not detailed here) indicate that subjects had a bias toward the center of the display. In other words, when an item appeared on the left side of a display, subjects were likely to estimate its location as being to the

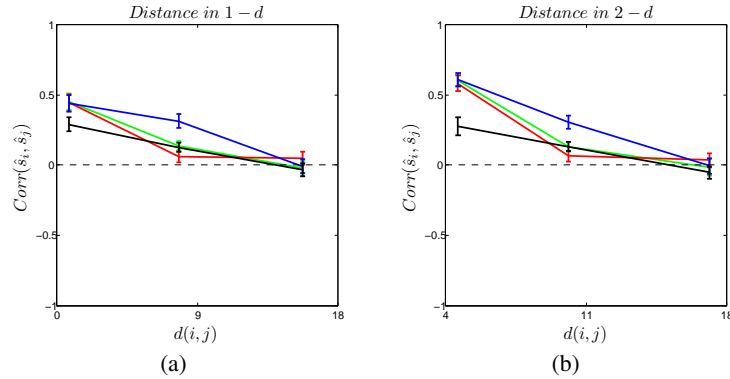

Figure 3: Subjects' mean correlation coefficients (and standard errors of the means) as a function of the distance $d(i,j)$ between items $i$ and $j$. $d(i,j)$ is measured either (a) in one dimension (considering only horizontal locations) or (b) in two dimensions (considering both horizontal and vertical locations). Each color corresponds to a different subject.

right of its actual location. Conversely, items appearing on the right side of a display were estimated as lying to the left of their actual locations. (This tendency can be observed in Figure 2a by noting that the black dots in this figure are often closer to the main diagonal than the magenta crosses). This bias is consistent with similar 'regression-to-the-mean' type biases previously reported in visual short-term memory for spatial frequency [5,8] and size [6].

Results for the experiment with three items were qualitatively similar. Figure 3 shows that, similar to the results observed in the experiment with two items, the magnitude of the correlations between subjects' position estimates decreases with Euclidean distance between items. In this figure, all $s_i$-$s_j$ pairs (recall that $s_i$ is the horizontal location of item $i$) for all display configurations were divided into 3 equal-length bins based on the Euclidean distance $d(i,j)$ between items $i$ and $j$ where we measured distance either in one dimension (considering only the horizontal locations of the items, Figure 3a) or in two dimensions (considering both horizontal and vertical locations, Figure 3b). Correlations differed significantly across different bins as indicated by one-way ANOVA (for both distance measures: $p < .01$ for all subjects, as well as for combined data from all subjects). Overall subjects exhibited a smaller number of significant $s_1$-$s_3$ correlations than $s_1$-$s_2$ or $s_2$-$s_3$ correlations. This is probably due to the fact that the $s_1$-$s_3$ pair had a larger vertical distance than the other pairs.

## 4 Explaining the covariances with correlated neural population responses

What could be the source of the specific form of covariances observed in our experiments? In this section we argue that dependencies of the form we observed in our experiments would naturally arise as a consequence of encoding multiple items in a population of neurons with correlated responses. To show this, we first consider encoding multiple stimuli with an idealized, correlated neural population and analytically derive an expression for the Fisher information matrix (FIM) in this model. This analytical expression for the FIM, in turn, predicts covariances of the type we observed in our experiments. We then simulate a more detailed and realistic network of spiking neurons and consider encoding and decoding the features of multiple items in this network. We show that this more realistic network also predicts covariances of the type we observed in our experiments. We emphasize that these predictions will be derived entirely from general properties of encoding and decoding information in correlated neural populations and as such do not depend on any specific assumptions about the properties of VSTM or how these properties might be implemented in neural populations.

**Encoding multiple stimuli in a neural population with correlated responses**

We first consider the problem of encoding $N$ stimuli ($\mathbf{s} = [s_1, \ldots, s_N]$) in a correlated population of $K$ neurons with Gaussian noise:

$$p(\mathbf{r}|\mathbf{s}) = \frac{1}{\sqrt{(2\pi)^K \det Q(\mathbf{s})}} \exp[-\frac{1}{2}(\mathbf{r} - \mathbf{f}(\mathbf{s}))^T Q^{-1}(\mathbf{s})(\mathbf{r} - \mathbf{f}(\mathbf{s}))] \tag{2}$$

where $\mathbf{r}$ is a vector containing the firing rates of the neurons in the population, $\mathbf{f}(\mathbf{s})$ represents the tuning functions of the neurons and $Q$ represents the specific covariance structure chosen. More specifically, we assume a 'limited range correlation structure' for $Q$ that has been analytically studied several times in the literature [9]–[15]. In a neural population with limited range correlations, the covariance between the firing rates of the $k$-th and $l$-th neurons (the $kl$-th cell of the covariance matrix) is assumed to be a monotonically decreasing function of the distance between their preferred stimuli [11]:

$$Q_{kl}(\mathbf{s}) = af_k(\mathbf{s})^\alpha f_l(\mathbf{s})^\alpha \exp(-\frac{||\mathbf{c}^{(k)} - \mathbf{c}^{(l)}||}{L}) \tag{3}$$

where $\mathbf{c}^{(k)}$ and $\mathbf{c}^{(l)}$ are the tuning function centers of the neurons. There is extensive experimental evidence for this type of correlation structure in the brain [16]-[19]. For instance, Zohary et al. [16] showed that correlations between motion direction selective MT neurons decrease with the difference in their preferred directions. This 'limited-range' assumption about the covariances between the firing rates of neurons will be crucial in explaining our experimental results in terms of the FIM of a correlated neural population encoding multiple stimuli.

We are interested in deriving the FIM, $J(\mathbf{s})$, for our correlated neural population encoding the stimuli $\mathbf{s}$. The significance of the FIM is that the inverse of the FIM provides a lower bound on the covariance matrix of any unbiased estimator of $\mathbf{s}$ and also expresses the asymptotic covariance matrix of the maximum-likelihood estimate of $\mathbf{s}$ in the limit of large $K$[1]. The $ij$-th cell of the FIM is defined as:

$$J_{ij}(\mathbf{s}) = -E[\frac{\partial^2}{\partial s_i \partial s_j} \log p(\mathbf{r}|\mathbf{s})] \tag{4}$$

Our derivation of $J(\mathbf{s})$ closely follows that of Wilke and Eurich in [11]. To derive an analytical expression for $J(\mathbf{s})$, we make a number of assumptions: (i) all neurons encode the same feature dimension (e.g. horizontal location in our experiment); (ii) indices of the neurons can be assigned such that neurons with adjacent indices have the closest tuning function centers; (iii) the centers of the tuning functions of neurons are linearly spaced with density $\eta$. The last two assumptions imply that the covariance between neurons with indices $k$ and $l$ can be expressed as $Q_{kl} = \rho^{|k-l|} a f_k^\alpha f_l^\alpha$ (we omitted the $\mathbf{s}$-dependence of $Q$ and $f$ for brevity) with $\rho = \exp(-1/(L\eta))$ where $L$ is a length parameter determining the spatial extent of the correlations. With these assumptions, it can be shown that (see Supplementary Material):

$$J_{ij}(\mathbf{s}) = \frac{1+\rho^2}{a(1-\rho^2)} \sum_{k=1}^{K} h_k^{(i)} h_k^{(j)} - \frac{2\rho}{a(1-\rho^2)} \sum_{k=1}^{K-1} h_k^{(i)} h_{k+1}^{(j)} + \frac{2\alpha^2}{1-\rho^2} \sum_{k=1}^{K} g_k^{(i)} g_k^{(j)} - \frac{2\alpha^2\rho^2}{1-\rho^2} \sum_{k=1}^{K-1} g_k^{(i)} g_{k+1}^{(j)} \tag{5}$$

where $h_k^{(i)} = \frac{1}{f_k^\alpha} \frac{\partial f_k}{\partial s_i}$ and $g_k^{(i)} = \frac{1}{f_k} \frac{\partial f_k}{\partial s_i}$.

Although not necessary for our results (see Supplementary Material), for convenience, we further assume that the neurons can be divided into $N$ groups where in each group the tuning functions are a function of the feature value of only one of the stimuli, i.e. $f_k(\mathbf{s}) = f_k(s_n)$ for neurons in group $n$, so that the effects of other stimuli on the mean firing rates of neurons in group $n$ are negligible. A population of neurons satisfying this assumption, as well as the assumptions (i)-(iii) above, for $N = 2$ is schematically illustrated in Figure 4a. We consider Gaussian tuning functions of the form: $f_k(s) = g \exp(-(s - c_k)^2/\sigma^2)$, with $c_k$ linearly spaced between $-12°$ and $12°$ and $g$ and $\sigma^2$ are assumed to be the same for all neurons. We take the inverse of $J(\mathbf{s})$, which provides a lower bound on the covariance matrix of any unbiased estimator of $\mathbf{s}$, and calculate correlation coefficients based on $J^{-1}(\mathbf{s})$ for each $\mathbf{s}$. For $N = 2$, for instance, we do this by calculating $J_{12}^{-1}(\mathbf{s})/\sqrt{J_{11}^{-1}(\mathbf{s})J_{22}^{-1}(\mathbf{s})}$. In Figure 4b, we plot this measure for all $s_1$, $s_2$ pairs between $-10°$ and $10°$. We see that the inverse of the FIM predicts correlations between the estimates of $s_1$ and $s_2$ and these correlations decrease with $|s_1 - s_2|$, just as we observed in our experiments (see Figure 4c). The best fits to experimental data were obtained with fairly broad tuning functions (see Figure 4 caption). For such broad tuning functions, the inverse of the FIM also predicts negative correlations when $|s_1 - s_2|$ is very large, which does not seem to be as strong in our data.

Intuitively, this result can be understood as follows. Consider the hypothetical neural population shown in Figure 4a encoding the pair $s_1$, $s_2$. In this population, it is assumed that $f_k(\mathbf{s}) = f_k(s_1)$

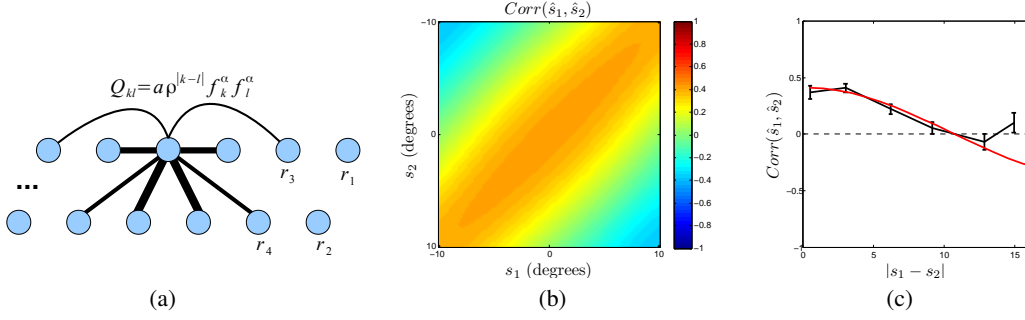

(a)                           (b)                           (c)

Figure 4: (a) A population of neurons satisfying all assumptions made in deriving the FIM. For neurons in the upper row $f_k(\mathbf{s}) = f_k(s_1)$, and for neurons in the lower row $f_k(\mathbf{s}) = f_k(s_2)$. The magnitude of correlations between two neurons is indicated by the thickness of the line connecting them. (b) Correlation coefficients estimated from the inverse of the FIM for all stimuli pairs $s_1$, $s_2$. (c) Mean correlation coefficients as a function of $|s_1 - s_2|$ (red: model's prediction; black: collapsed data from all 4 subjects in the experiment with $N = 2$). Parameters: $\alpha = 0.5$, $g = 50$, $a = 1$ (these were set to biologically plausible values); other parameters: $K = 500$, $\sigma = 9.0$, $L = 0.0325$ (the last two were chosen to provide a good fit to the experimental results).

for neurons in the upper row, and $f_k(\mathbf{s}) = f_k(s_2)$ for neurons in the lower row. Suppose that in the upper row, the $k$-th neuron has the best-matching tuning function for a given $s_1$. Therefore, on average, the $k$-th neuron has the highest firing rate in response to $s_1$. However, since the responses of the neurons are stochastic, on some trials, neurons to the left (right) of the $k$-th neuron will have the highest firing rate in response to $s_1$. When this happens, neurons in the lower row with similar preferences will be more likely to get activated, due to the limited-range correlations between the neurons. This, in turn, will introduce correlations in an estimator of $\mathbf{s}$ based on $\mathbf{r}$ that are strongest when the absolute difference between $s_1$ and $s_2$ is small.

**Encoding and decoding multiple stimuli in a network of spiking neurons**

There might be two concerns about the analytical argument given in the previous subsection. The first is that we needed to make many assumptions in order to derive an analytic expression for $J(\mathbf{s})$. It is not clear if we would get similar results when one or more of these assumptions are violated. Secondly, the interpretation of the off-diagonal terms (covariances) in $J^{-1}(\mathbf{s})$ is somewhat different from the interpretation of the diagonal terms (variances). Although the diagonal terms provide lower bounds on the variances of any unbiased estimator of $\mathbf{s}$, the off-diagonal terms do not necessarily provide lower bounds on the covariances of the estimates, that is, there might be estimators with lower covariances.

To address these concerns, we simulated a more detailed and realistic network of spiking neurons. The network consisted of two layers. In the input layer, there were 169 Poisson neurons arranged in a $13 \times 13$ grid with linearly spaced receptive field centers between $-12°$ and $12°$ along both horizontal and vertical directions. On a given trial, the firing rate of the $k$-th input neuron was determined by the following equation:

$$r_k = g_{in}[\exp(-\frac{\|\mathbf{x}_1 - \mathbf{c}^{(k)}\|}{\sigma_{in}}) + \exp(-\frac{\|\mathbf{x}_2 - \mathbf{c}^{(k)}\|}{\sigma_{in}})] \tag{6}$$

for the case of $N = 2$. Here $\|\cdot\|$ is the Euclidean norm, $\mathbf{x}_i$ is the vertical and horizontal locations of the $i$-th stimulus, $\mathbf{c}^{(k)}$ is the receptive field center of the input neuron, $g_{in}$ is a gain parameter and $\sigma_{in}$ is a scale parameter (both assumed to be the same for all input neurons).

The output layer consisted of simple leaky integrate-and-fire neurons. There were 169 of these neurons arranged in a $13 \times 13$ grid with the receptive field center of each neuron matching the receptive field center of the corresponding neuron in the input layer. We induced limited-range correlations between the output neurons through receptive field overlap, although other ways of introducing limited-range correlations can be considered such as through local lateral connections. Each output neuron had a Gaussian connection weight profile centered at the corresponding input

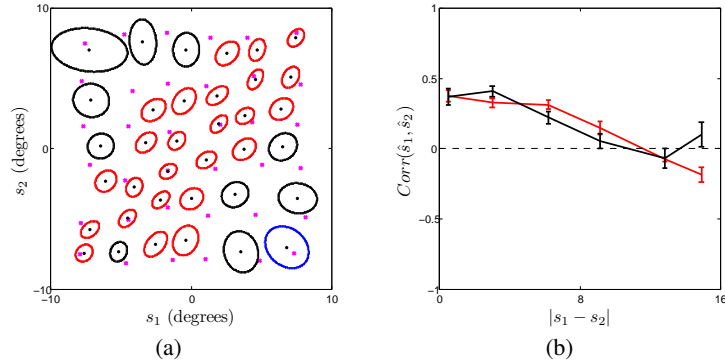

Figure 5: (a) Results for the network model. The actual display configurations **s** are represented by magenta dots, the estimated means based on the model's responses are represented by black dots and the estimated covariances are represented by contours (with red contours representing $\Sigma(\mathbf{s})$ for which the two dimensions were significantly correlated at the $p < 0.05$ level). (b) The mean correlation coefficients (and standard errors of the means) as a function of $|s_1 - s_2|$ (red: model prediction; black: collapsed data from all 4 subjects in the experiment with $N = 2$). Model parameters: $g_{in} = 120$, $\sigma_{in} = 2$, $\sigma_{out} = 2$. Parameters were chosen to provide a good fit to the experimental results.

neuron and with a standard deviation of $\sigma_{out}$. The output neurons had a threshold of -55 mV and a reset potential of -70 mV. Each spike of an input neuron $k$ instantaneously increased the voltage of an output neuron $l$ by $10w_{kl}$ mV, where $w_{kl}$ is the connection weight between the two neurons and the voltage decayed with a time constant of 10 ms. We implemented the network in Python using the Brian neural network simulator [20].

We simulated this network with the same display configurations presented to our subjects in the experiment with $N = 2$. Each of the 36 configurations was presented 96 times to the network, yielding a total of 3456 trials. For each trial, the network was simulated for 100 ms and its estimates of $s_1$ and $s_2$ were read out using a suboptimal decoding strategy. Specifically, to get an estimate of $s_1$, we considered only the row of neurons in the output layer whose preferred vertical locations were closest to the vertical location of the first stimulus and then we fit a Gaussian function (with amplitude, peak location and width parameters) to the activity profile of this row of neurons and considered the estimated peak location as the model's estimate of $s_1$. We did the same for obtaining an estimate of $s_2$. Figure 5 shows the results for the network model. Similar to our experimental results, the spiking network model predicts correlations between the estimates of $s_1$ and $s_2$ and these correlations decrease with $|s_1 - s_2|$ (correlations differed significantly across different bins as indicated by a one-way ANOVA: $F(5, 30) = 22.9713$, $p < 10^{-8}$; see Figure 5b). Interestingly, the model was also able to replicate the biases toward the center of the screen observed in the experimental data. This is due to the fact that output neurons near the center of the display tended to have higher activity levels, since they have more connections with the input neurons compared to the output neurons near the edges of the display.

## 5 Discussion

Properties of correlations among the responses of neural populations have been studied extensively from both theoretical and experimental perspectives. However, the implications of these correlations for jointly encoding multiple items in memory are not known. Our results here suggest that one consequence of limited-range neural correlations might be correlations in the estimates of the feature values of different items that decrease with the absolute difference between their feature values. An interesting question is whether our results generalize to other feature dimensions, such as orientation, spatial frequency etc. Preliminary data from our lab suggest that covariances of the type reported here for spatial location might also be observed in VSTM for orientation.

**Acknowledgments:** We thank R. Moreno-Bote for helpful discussions. This work was supported by a research grant from the National Science Foundation (DRL-0817250).

## Footnotes

[1] $J^{-1}(\mathbf{s})$ provides a lower bound on the covariance matrix of any unbiased estimator of $\mathbf{s}$ in the matrix sense (where $A \geq B$ means $A - B$ is positive semi-definite).

# References

[1] Bays, P.M. & Husain, M. (2008) Dynamic shifts of limited working memory resources in human vision. *Science* 321:851-854.

[2] Zhang, P.H. & Luck, S.J. (2008) Discrete fixed-resolution representations in visual working memory. *Nature* 453:233-235.

[3] Jiang, Y., Olson, I.R. & Chun, M.M. (2000) Organization of visual short-term memory. *Journal of Experimental Psychology: Learning, Memory and Cognition* **26**(3):683-702.

[4] Kahana, M.J. & Sekuler, R. (2002) Recognizing spatial patterns: a noisy exemplar approach. *Vision Research* 42:2177-2192.

[5] Huang, J. & Sekuler, R. (2010) Distortions in recall from visual memory: Two classes of attractors at work *Journal of Vision* 10:1-27.

[6] Brady, T.F. & Alvarez, G.A. (in press) Hierarchical encoding in visual working memory: ensemble statistics bias memory for individual items. *Psychological Science*.

[7] Rasmussen, C.E. & Williams, C.K.I (2006) *Gaussian Processes for Machine Learning.* MIT Press.

[8] Ma, W.J. & Wilken, P. (2004) A detection theory account of change detection. *Journal of Vision* 4:1120-1135.

[9] Abbott, L.F. & Dayan, P. (1999) The effect of correlated variability on the accuracy of a population code. *Neural Computation* 11:91-101.

[10] Shamir, M. & Sompolinsky, H. (2004) Nonlinear population codes. *Neural Computation* 16:1105-1136.

[11] Wilke, S.D. & Eurich, C.W. (2001) Representational accuracy of stochastic neural populations. *Neural Computation* 14:155-189.

[12] Berens, P., Ecker, A.S., Gerwinn, S., Tolias, A.S. & Bethge, M. (2011) Reassessing optimal neural population codes with neurometric functions. *PNAS* **108**(11): 44234428.

[13] Snippe, H.P. & Koenderink, J.J. (1992) Information in channel-coded systems: correlated receivers. *Biological Cybernetics* 67: 183-190.

[14] Sompolinsky, H., Yoon, H., Kang, K. & Shamir, M. (2001) Population coding in neural systems with correlated noise. *Physical Review E* 64: 051904.

[15] Josić, K., Shea-Brown, E., Doiron, B. & de la Rocha, J. (2009) Stimulus-dependent correlations and population codes. *Neural Computation* 21:2774-2804.

[16] Zohary, E., Shadlen, M.N. & Newsome, W.T. (1994) Correlated neuronal discharge rate and its implications for psychophysical performance. *Nature* 370:140-143.

[17] Bair, W., Zohary, E. & Newsome, W.T. (2001) Correlated firing in macaque area MT: Time scales and relationship to behavior. *The Journal of Neuroscience* **21**(5): 16761697.

[18] Maynard, E.M., Hatsopoulos, N.G., Ojakangas, C.L., Acuna, B.D., Sanes, J.N., Norman, R.A. & Donoghue, J.P. (1999) Neuronal interactions improve cortical population coding of movement direction. *The Journal of Neuroscience* **19**(18): 80838093.

[19] Smith, M.A. & Kohn, A. (2008) Spatial and temporal scales of neuronal correlation in primary visual cortex. *The Journal of Neuroscience* **28**(48): 1259112603.

[20] Goodman, D. & Brette, R. (2008) Brian: a simulator for spiking neural networks in Python. *Frontiers in Neuroinformatics* **2**:5. doi: 10.3389/neuro.11.005.2008.

